# Network Generality, Training Required, and Precision Required

John S. Denker and Ben S. Wittner [1]
AT&T Bell Laboratories
Holmdel, New Jersey 07733

> Keep your hand on your wallet.
> — Leon Cooper, 1987

## Abstract

We show how to estimate (1) the number of functions that can be implemented by a particular network architecture, (2) how much analog precision is needed in the connections in the network, and (3) the number of training examples the network must see before it can be expected to form reliable generalizations.

## Generality versus Training Data Required

Consider the following objectives: First, the network should be very powerful and versatile, i.e., it should implement any function (truth table) you like, and secondly, it should learn easily, forming meaningful generalizations from a small number of training examples. Well, it is information-theoretically impossible to create such a network. We will present here a simplified argument; a more complete and sophisticated version can be found in Denker et al. (1987).

It is customary to regard learning as a dynamical process: adjusting the weights (etc.) in a single network. In order to derive the results of this paper, however, we take a different viewpoint, which we call the ensemble viewpoint. Imagine making a very large number of replicas of the network. Each replica has the same architecture as the original, but the weights are set differently in each case. No further adjustment takes place; the "learning process" consists of winnowing the ensemble of replicas, searching for the one(s) that satisfy our requirements.

Training proceeds as follows: We present each item in the training set to every network in the ensemble. That is, we use the abscissa of the training pattern as input to the network, and compare the ordinate of the training pattern to see if it agrees with the actual output of the network. For each network, we keep a score reflecting how many times (and how badly) it disagreed with a training item. Networks with the lowest score are the ones that agree best with the training data. If we had complete confidence in

the reliability of the training set, we could at each step simply throw away all networks that disagree.

For definiteness, let us consider a typical network architecture, with $N_0$ input wires and $N_l$ units in each processing layer $l$, for $l \in \{1 \cdots L\}$. For simplicity we assume $N_L = 1$. We recognize the importance of networks with continuous-valued inputs and outputs, but we will concentrate for now on training (and testing) patterns that are discrete, with $N \equiv N_0$ bits of abscissa and $N_L = 1$ bit of ordinate. This allows us to classify the networks into bins according to what Boolean input-output relation they implement, and simply consider the ensemble of bins.

There are $2^{2^N}$ possible bins. If the network architecture is completely general and powerful, all $2^{2^N}$ functions will exist in the ensemble of bins. On average, one expects that each training item will throw away at most half of the bins. Assuming maximal efficiency, if $m$ training items are used, then when $m \gtrsim 2^N$ there will be only one bin remaining, and that must be the unique function that consistently describes all the data. But there are only $2^N$ possible abscissas using $N$ bits. Therefore a truly general network cannot possibly exhibit meaningful generalization — 100% of the possible data is needed for training.

Now suppose that the network is not completely general, so that even with all possible settings of the weights we can only create functions in $2^{S_0}$ bins, where $S_0 \ll 2^N$. We call $S_0$ the initial entropy of the network. A more formal and general definition is given in Denker et al. (1987). Once again, we can use the training data to winnow the ensemble, and when $m \gtrsim S_0$, there will be only one remaining bin. That function will presumably generalize correctly to the remaining $2^N - m$ possible patterns. Certainly that function is the best we can do with the network architecture and the training data we were given.

The usual problem with automatic learning is this: If the network is too general, $S_0$ will be large, and an inordinate amount of training data will be required. The required amount of data may be simply unavailable, or it may be so large that training would be prohibitively time-consuming. The shows the critical importance of building a network that is not more general than necessary.

## Estimating the Entropy

In real engineering situations, it is important to be able to estimate the initial entropy of various proposed designs, since that determines the amount of training data that will be required. Calculating $S_0$ directly from the definition is prohibitively difficult, but we can use the definition to derive useful approximate expressions. (You wouldn't want to calculate the thermodynamic entropy of a bucket of water directly from the definition, either.)

Suppose that the weights in the network at each connection $i$ were not continuously adjustable real numbers, but rather were specified by a discrete code with $b_i$ bits. Then the total number of bits required to specify the configuration of the network is

$$B = \sum_i b_i \tag{1}$$

Now the total number of functions that could possibly be implemented by such a network architecture would be at most $2^B$. The actual number will always be smaller than this, since there are various ways in which different settings of the weights can lead to identical functions (bins). For one thing, for each hidden layer $l \in \{1 \cdots L-1\}$, the numbering of the hidden units can be permuted, and the polarity of the hidden units can be flipped, which means that $2^{S_0}$ is less than $2^B$ by a factor (among others) of $\prod_l N_l! \, 2^{N_l}$. In addition, if there is an inordinately large number of bits $b_i$ at each connection, there will be many settings where small changes in the connection will be immaterial. This will make $2^{S_0}$ smaller by an additional factor. We expect $\partial S_0 / \partial b_i \approx 1$ when $b_i$ is small, and $\partial S_0 / \partial b_i \approx 0$ when $b_i$ is large; we must now figure out where the crossover occurs.

The number of "useful and significant" bits of precision, which we designate $b^*$, typically scales like the logarithm of number of connections to the unit in question. This can be understood as follows: suppose there are $N$ connections into a given unit, and an input signal to that unit of some size $A$ is observed to be significant (the exact value of $A$ drops out of the present calculation). Then there is no point in having a weight with magnitude much larger than $A$, nor much smaller than $A/N$. That is, the dynamic range should be comparable to the number of connections. (This argument is not exact, and it is easy to devise exceptions, but the conclusion remains useful.) If only a fraction $1/S$ of the units in the previous layer are active (nonzero) at a time, the needed dynamic range is reduced. This implies $b^* \approx \log(N/S)$.

Note: our calculation does not involve the dynamics of the learning process. Some numerical methods (including versions of back propagation) commonly require a number of temporary "guard bits" on each weight, as pointed out by Richard Durbin (private communication). Another $\log N$ bits ought to suffice. These bits are not needed after learning is complete, and do not contribute to $S_0$.

If we combine these ideas and apply them to a network with $N$ units in each layer, fully connected, we arrive at the following expression for the number of different Boolean functions that can be implemented by such a network:

$$2^{S_0} \approx \frac{2^B}{N! \, 2^N} \tag{2}$$

where

$$B \approx L N^2 \log N \tag{3}$$

These results depend on the fact that we are considering only a very restricted type of processing unit: the output is a monotone function of a weighted sum of inputs. Cover

(1965) discussed in considerable depth the capabilities of such units. Valiant (1986) has explored the learning capabilities of various models of computation.

Abu-Mustafa has emphasized the principles of information and entropy and applied them to measuring the properties of the training set. At this conference, formulas similar to equation 3 arose in the work of Baum, Psaltis, and Venkatesh, in the context of calculating the number of different training patterns a network should be *able to* memorize. We originally proposed equation 2 as an estimate of the number of patterns the network would *have to* memorize before it could form a reliable generalization. The basic idea, which has numerous consequences, is to estimate the number of (bins of) networks that can be realized.

## Footnotes

[1] Currently at NYNEX Science and Technology, 500 Westchester Ave., White Plains, NY 10604

# References

1. Yasser Abu-Mustafa, these proceedings.

2. Eric Baum, these proceedings.

3. T. M. Cover, "Geometrical and statistical properties of systems of linear inequalities with applications in pattern recognition," *IEEE Trans. Elec. Comp.*, EC-14, 326-334, (June 1965)

4. John Denker, Daniel Schwartz, Ben Wittner, Sara Solla, John Hopfield, Richard Howard, and Lawrence Jackel, Complex Systems, in press (1987).

5. Demetri Psaltis, these proceedings.

6. L. G. Valiant, SIAM J. Comput. **15(2)**, 531 (1986), and references therein.

7. Santosh Venkatesh, these proceedings.
